# Replacing supervised classification learning by Slow Feature Analysis in spiking neural networks

**Stefan Klampfl, Wolfgang Maass**
Institute for Theoretical Computer Science
Graz University of Technology
A-8010 Graz, Austria
{klampfl,maass}@igi.tugraz.at

## Abstract

It is open how neurons in the brain are able to learn without supervision to discriminate between spatio-temporal firing patterns of presynaptic neurons. We show that a known unsupervised learning algorithm, Slow Feature Analysis (SFA), is able to acquire the classification capability of Fisher's Linear Discriminant (FLD), a powerful algorithm for supervised learning, if temporally adjacent samples are likely to be from the same class. We also demonstrate that it enables linear readout neurons of cortical microcircuits to learn the detection of repeating firing patterns within a stream of spike trains with the same firing statistics, as well as discrimination of spoken digits, in an unsupervised manner.

## 1 Introduction

Since the presence of supervision in biological learning mechanisms is rare, organisms often have to rely on the ability of these mechanisms to extract statistical regularities from their environment. Recent neurobiological experiments [1] have suggested that the brain uses some type of slowness objective to learn the categorization of external objects without a supervisor. Slow Feature Analysis (SFA) [2] could be a possible mechanism for that. We establish a relationship between the unsupervised SFA learning method and a commonly used method for supervised classification learning: Fisher's Linear Discriminant (FLD) [3]. More precisely, we show that SFA approximates the classification capability of FLD by replacing the supervisor with the simple heuristics that two temporally adjacent samples in the input time series are likely to be from the same class. Furthermore, we demonstrate in simulations of a cortical microcircuit model that SFA could also be an important ingredient in extracting temporally stable information from trajectories of network states and that it supports the idea of "anytime" computing, i.e., it provides information about the stimulus identity not only at the end of a trajectory of network states, but already much earlier.

This paper is structured as follows. We start in section 2 with brief recaps of the definitions of SFA and FLD. We discuss the relationship between these methods for unsupervised and supervised learning in section 3, and investigate the application of SFA to trajectories in section 4. In section 5 we report results of computer simulations of several SFA readouts of a cortical microcircuit model. Section 6 concludes with a discussion.

## 2 Basic Definitions

### 2.1 Slow Feature Analysis (SFA)

Slow Feature Analysis (SFA) [2] is an *unsupervised* learning algorithm that extracts the slowest components $y_i$ from a multi-dimensional input time series $\mathbf{x}$ by minimizing the temporal variation

$\Delta(y_i)$ of the output signal $y_i$, which is defined in [2] as the average of its squared temporal derivative. Thus the objective is to minimize

$$\min \quad \Delta(y_i) := \langle \dot{y_i}^2 \rangle_t. \tag{1}$$

The notation $\langle \cdot \rangle_t$ denotes averaging over time, and $\dot{y}$ is the time derivative of $y$. The additional constraints of zero mean ($\langle y_i \rangle_t = 0$) and unit variance ($\langle y_i^2 \rangle_t = 1$) avoid the trivial constant solution $y_i(t) \equiv 0$. If multiple slow features are extracted, a third constraint ($\langle y_i y_j \rangle_t = 0, \forall j < i$) ensures that they are decorrelated and ordered by decreasing slowness, i.e., $y_1$ is the slowest feature extracted, $y_2$ the second slowest feature, and so on. In other words, SFA finds those functions $g_i$ out of a certain predefined function space that produce the slowest possible outputs $y_i = g_i(\mathbf{x})$ under these constraints.

This optimization problem is hard to solve in the general case [4], but if we assume that the time series $\mathbf{x}$ has zero mean ($\langle \mathbf{x} \rangle_t = \mathbf{0}$) and if we only allow linear functions $y = \mathbf{w}^T \mathbf{x}$ the problem simplifies to the objective

$$\min \quad J_{SFA}(\mathbf{w}) := \frac{\mathbf{w}^T \langle \dot{\mathbf{x}}\dot{\mathbf{x}}^T \rangle_t \mathbf{w}}{\mathbf{w}^T \langle \mathbf{x}\mathbf{x}^T \rangle_t \mathbf{w}}. \tag{2}$$

The matrix $\langle \mathbf{x}\mathbf{x}^T \rangle_t$ is the covariance matrix of the input time series and $\langle \dot{\mathbf{x}}\dot{\mathbf{x}}^T \rangle_t$ denotes the covariance matrix of time derivatives (or time differences, for discrete time) of the input time series. The weight vector $\mathbf{w}$ which minimizes (2) is the solution to the generalized eigenvalue problem

$$\langle \dot{\mathbf{x}}\dot{\mathbf{x}}^T \rangle_t \mathbf{w} = \lambda \langle \mathbf{x}\mathbf{x}^T \rangle_t \mathbf{w} \tag{3}$$

corresponding to the *smallest* eigenvalue $\lambda$. To make use of a larger function space one typically considers linear combinations $y = \mathbf{w}^T \mathbf{z}$ of fixed nonlinear expansions $\mathbf{z} = \mathbf{h}(\mathbf{x})$ and performs the optimization (2) in this high-dimensional space.

## 2.2 Fisher's Linear Discriminant (FLD)

Fisher's Linear Discriminant (FLD) [3], on the other hand, is a *supervised* learning method, since it is applied to *labeled* training examples $\langle \mathbf{x}, c \rangle$, where $c \in \{1, \ldots, C\}$ is the class to which this example $\mathbf{x}$ belongs. The goal is to find a weight vector $\mathbf{w}$ so that the ability to predict the class of $\mathbf{x}$ from the value of $\mathbf{w}^T \mathbf{x}$ is maximized.

FLD searches for that projection direction $\mathbf{w}$ which maximizes the separation between classes while at the same time minimizing the variance within classes, thereby minimizing the class overlap of the projected values:

$$\max \quad J_{FLD}(\mathbf{w}) := \frac{\mathbf{w}^T \mathbf{S}_B \mathbf{w}}{\mathbf{w}^T \mathbf{S}_W \mathbf{w}}. \tag{4}$$

For $C$ point sets $S_c$, each with $N_c$ elements and means $\boldsymbol{\mu}_c$, $\mathbf{S}_B$ is the between-class covariance matrix given by the separation of the class means, $\mathbf{S}_B = \sum_c N_c (\boldsymbol{\mu}_c - \boldsymbol{\mu})(\boldsymbol{\mu}_c - \boldsymbol{\mu})^T$, and $\mathbf{S}_W$ is the within-class covariance matrix given by $\mathbf{S}_W = \sum_c \sum_{\mathbf{x} \in S_c} (\mathbf{x} - \boldsymbol{\mu}_c)(\mathbf{x} - \boldsymbol{\mu}_c)^T$. Again, the vector $\mathbf{w}$ optimizing (4) can be viewed as the solution to a generalized eigenvalue problem,

$$\mathbf{S}_B \mathbf{w} = \lambda \mathbf{S}_W \mathbf{w}, \tag{5}$$

corresponding to the *largest* eigenvalue $\lambda$.

# 3 SFA can acquire the classification capability of FLD

SFA and FLD receive different data types as inputs: unlabeled time series for SFA, in contrast to labeled single data points for the FLD. Therefore, in order to apply the unsupervised SFA learning algorithm to the same classification problem as the supervised FLD, we have to convert the labeled training samples into a time series of unlabeled data points that can serve as an input to the SFA algorithm[1]. In the following we investigate the relationship between the weight vectors found by both methods for a particular way of time series generation.

We consider a classification problem with $C$ classes, i.e., assume we are given point sets $S_1, S_2, \ldots, S_C \subset \mathbb{R}^n$. Let $N_c$ be the number of points in $S_c$ and let $N = \sum_{c=1}^{C} N_c$ be the total number of points. In order to create a time series $\mathbf{x}_t$ out of these point sets we define a Markov model with $C$ states $S = \{1, 2, \ldots, C\}$, one for each class, and choose at each time step $t = 1, \ldots, T$ a random point from the class that corresponds to the current state in the Markov model. We define the transition probability from state $i \in S$ to state $j \in S$ as

$$P_{ij} = \begin{cases} a \cdot \frac{N_j}{N} & \text{if } i \neq j, \\ 1 - \sum_{k \neq j} P_{ik} & \text{if } i = j, \end{cases} \tag{6}$$

with some appropriate constant $a > 0$. The stationary distribution of this Markov model is $\boldsymbol{\pi} = (N_1/N, N_2/N, \ldots, N_C/N)$. We choose the initial distribution $\mathbf{p}_0 = \boldsymbol{\pi}$, i.e., at any time $t$ the probability that point $\mathbf{x}_t$ is chosen from class $c$ is $N_c/N$.

For this particular way of generating the time series from the original classification problem we can express the matrices $\langle \mathbf{x}\mathbf{x}^T \rangle_t$ and $\langle \dot{\mathbf{x}}\dot{\mathbf{x}}^T \rangle_t$ of the SFA objective (2) in terms of the within-class and between-class scatter matrices of the FLD (4), $\mathbf{S}_W$ and $\mathbf{S}_B$, in the following way [6]:

$$\langle \mathbf{x}\mathbf{x}^T \rangle_t = \frac{1}{N}\mathbf{S}_W + \frac{1}{N}\mathbf{S}_B \tag{7}$$

$$\langle \dot{\mathbf{x}}\dot{\mathbf{x}}^T \rangle_t = \frac{2}{N}\mathbf{S}_W + a \cdot \frac{2}{N}\mathbf{S}_B \tag{8}$$

Note that only $\langle \dot{\mathbf{x}}\dot{\mathbf{x}}^T \rangle_t$ depends on $a$, whereas $\langle \mathbf{x}\mathbf{x}^T \rangle_t$ does not.

For small $a$ we can neglect the effect of $\mathbf{S}_B$ on $\langle \dot{\mathbf{x}}\dot{\mathbf{x}}^T \rangle_t$ in (8). In this case the time series consists mainly of transitions within a class, whereas switching between the two classes is relatively rare. Therefore the covariance of time derivatives is mostly determined by the within-class scatter of the two point sets, and both matrices become approximately proportional: $\langle \dot{\mathbf{x}}\dot{\mathbf{x}}^T \rangle_t \approx 2/N \cdot \mathbf{S}_W$. Moreover, if we assume that $\mathbf{S}_W$ (and therefore $\langle \dot{\mathbf{x}}\dot{\mathbf{x}}^T \rangle_t$) is positive definite, we can rewrite the SFA objective (2) as

$$\min J_{SFA}(\mathbf{w}) \Leftrightarrow \max \frac{1}{J_{SFA}(\mathbf{w})} \Leftrightarrow \max \frac{\mathbf{w}^T \langle \mathbf{x}\mathbf{x}^T \rangle_t \mathbf{w}}{\mathbf{w}^T \langle \dot{\mathbf{x}}\dot{\mathbf{x}}^T \rangle_t \mathbf{w}}$$

$$\Leftrightarrow \max \frac{1}{2} + \frac{1}{2} \cdot \frac{\mathbf{w}^T \mathbf{S}_B \mathbf{w}}{\mathbf{w}^T \mathbf{S}_W \mathbf{w}} \Leftrightarrow \max J_{FLD}(\mathbf{w}). \tag{9}$$

That is, the weight vector that optimizes the SFA objective (2) also optimizes the FLD objective (4). For $C > 2$ this equivalence can be seen by recalling the definition of SFA as a generalized eigenvalue problem (3) and inserting (7) and (8):

$$\langle \dot{\mathbf{x}}\dot{\mathbf{x}}^T \rangle_t \mathbf{W} = \langle \mathbf{x}\mathbf{x}^T \rangle_t \mathbf{W}\boldsymbol{\Lambda}$$

$$\mathbf{S}_B \mathbf{W} = \mathbf{S}_W \mathbf{W} \left[ 2\boldsymbol{\Lambda}^{-1} - \mathbf{E} \right], \tag{10}$$

where $\mathbf{W} = (\mathbf{w}_1, \ldots, \mathbf{w}_n)$ is the matrix of generalized eigenvectors and $\boldsymbol{\Lambda} = \mathrm{diag}(\lambda_1, \ldots, \lambda_n)$ is the diagonal matrix of generalized eigenvalues. The last line of (10) is just the formulation of FLD as a generalized eigenvalue problem (5). More precisely, the eigenvectors of the SFA problem are also eigenvectors of the FLD problem. Note that the eigenvalues correspond by $\lambda_i^{FLD} = 2/\lambda_i^{SFA} - 1$, which means the order of eigenvalues is reversed ($\lambda_i^{SFA} > 0$). Thus, the subspace spanned by the slowest features is the same that optimizes separability in terms of Fisher's Discriminant, and the slowest feature is the weight vector which achieves maximal separation.

Figure 1A demonstrates this relationship on a sample two-class problem in two dimensions for the special case of $N_1 = N_2 = N/2$. In this case at each time the class is switched with probability $p = a/2$ or is left unchanged with probability $1 - p$. We interpret the weight vectors found by both methods as normal vectors of hyperplanes in the input space, which we place simply onto the mean value $\boldsymbol{\mu}$ of all training data points (i.e., the hyperplanes are defined as $\mathbf{w}^T \mathbf{x} = \theta$ with $\theta = \mathbf{w}^T \boldsymbol{\mu}$). One sees that the weight vector found by the application of SFA to the training time series $\mathbf{x}_t$ generated with $p = 0.2$ is approximately equal to the weight vector resulting from FLD on the initial sets of training points. This demonstrates that SFA has extracted the class of the points as the slowest varying feature by finding a direction that separates both classes.

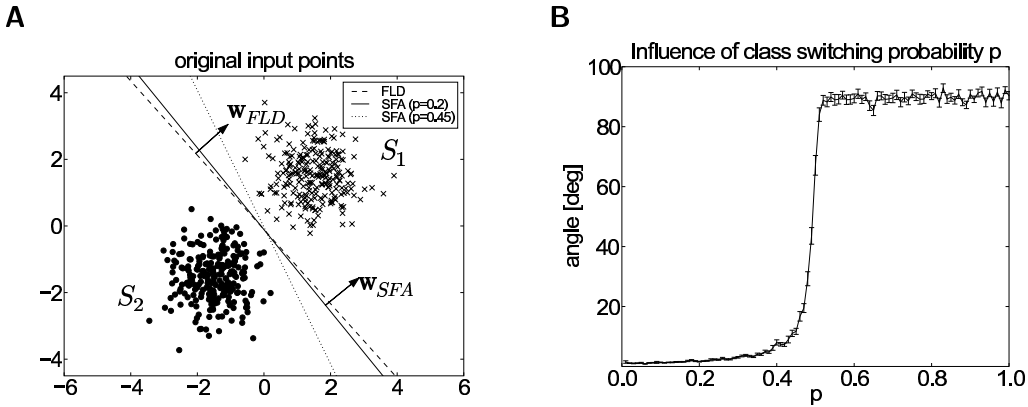

Figure 1: Relationship between SFA and FLD for a two-class problem in 2D. (**A**) Sample point sets with 250 points for each class. The dashed line indicates a hyperplane corresponding to the weight vector $\mathbf{w}_{FLD}$ resulting from the application of FLD to the two-class problem. The black solid line shows a hyperplane for the weight vector $\mathbf{w}_{SFA}$ resulting from SFA applied to the time series generated from these training points as described in the text ($T = 5000$, $p = 0.2$). The dotted line displays an additional SFA hyperplane resulting from a time series generated with $p = 0.45$. All hyperplanes are placed onto the mean value of all training points. (**B**) Dependence of the error between the weight vectors found by FLD and SFA on the switching probability $p$. This error is defined as the average angle between the weight vectors obtained on 100 randomly chosen classification problems. Error bars denote the standard error of the mean.

Figure 1B quantifies the deviation of the weight vector resulting from the application of SFA to the time series from the one found by FLD on the original points. We use the average angle between both weight vectors as an error measure. It can be seen that if $p$ is low, i.e., transitions between classes are rare compared to transitions within a class, the angle between the vectors is small and SFA approximates FLD very well. The angle increases moderately with increasing $p$; even with higher values of $p$ (up to 0.45) the approximation is reasonable and a good classification by the slowest feature can be achieved (see dotted hyperplane in Figure 1A). As soon as $p$ reaches a value of about 0.5, the error grows almost immediately to the maximal value of $90°$. For $p = 0.5$ ($a = 1$) points are chosen independently of their class, making the matrices $\langle \dot{\mathbf{x}}\dot{\mathbf{x}}^T \rangle_t$ and $\langle \mathbf{x}\mathbf{x}^T \rangle_t$ proportional. This means that every possible vector $\mathbf{w}$ is a solution to the generalized eigenvalue problem (3), resulting in an average angle of about $45°$.

## 4   Application to trajectories of training examples

In the previous section we have shown that SFA approximates the classification capability of FLD if the probability is low that two successive points in the input time series to SFA are from different classes. Apart from this temporal structure induced by the class information, however, these samples are chosen independently at each time step. In this section we investigate how the SFA objective changes when the input time series consists of a sequence of *trajectories* of samples instead of individual points only.

First, we consider a time series $\mathbf{x}_t$ consisting of multiple repetitions of a fixed predefined trajectory $\tilde{\mathbf{t}}$, which is embedded into noise input consisting of a random number of points drawn from the same distribution as the trajectory points, but independently at each time step. It is easy to show [6] that for such a time series the SFA objective (2) reduces to finding the eigenvector of the matrix $\tilde{\boldsymbol{\Sigma}}_t$ corresponding to the largest eigenvalue. $\tilde{\boldsymbol{\Sigma}}_t$ is the covariance matrix of the trajectory $\tilde{\mathbf{t}}$ with $\tilde{\mathbf{t}}$ delayed by one time step, i.e., it measures the temporal covariances (hence the index $t$) of $\tilde{\mathbf{t}}$ with time lag 1. Since the transitions between two successive points of the trajectory $\tilde{\mathbf{t}}$ occur much more often in the time series $\mathbf{x}_t$ than transitions between any other possible pair of points, SFA has to respond as smoothly as possible (i.e., maximize the temporal correlations) during $\tilde{\mathbf{t}}$ in order to produce the

slowest possible output. This means that SFA is able to detect repetitions of $\tilde{\mathbf{t}}$ by responding during such instances with a distinctive shape.

Next, we consider a classification problem given by $C$ sets of trajectories, $\mathcal{T}_1, \mathcal{T}_2, \ldots, \mathcal{T}_C \subset (\mathbb{R}^n)^{\tilde{T}}$, i.e., the elements of each set $\mathcal{T}_c$ are sequences of $\tilde{T}$ $n$-dimensional points. We generate a time series according to the same Markov model as described in the previous section, except that we do not choose individual points at each time step, rather we generate a sequence of trajectories. For this time series we can express the matrices $\langle \mathbf{xx}^T \rangle_t$ and $\langle \dot{\mathbf{x}}\dot{\mathbf{x}}^T \rangle_t$ in terms of the within-class and between-class scatter of the individual points of the trajectories in $\mathcal{T}_c$, analogously to (7) and (8) [6]. While the expression for $\langle \mathbf{xx}^T \rangle_t$ is unchanged the temporal correlations induced by the use of trajectories however have an effect on the covariance of temporal differences $\langle \dot{\mathbf{x}}\dot{\mathbf{x}}^T \rangle_t$. First, this matrix additionally depends on the temporal covariance $\tilde{\mathbf{\Sigma}}_t$ with time lag 1 of all available trajectories in all sets $\mathcal{T}_c$. Second, the effective switching probability is reduced by a factor of $1/\tilde{T}$. Whenever a trajectory is selected, $\tilde{T}$ points from the same class are presented in succession.

This means that even for a small switching probability[2] the objective of SFA cannot be solely reduced to the FLD objective, but rather that there is a trade-off between the tendency to separate trajectories of different classes (as explained by the relation between $\mathbf{S}_B$ and $\mathbf{S}_W$) and the tendency to produce smooth responses during individual trajectories (determined by the temporal covariance matrix $\tilde{\mathbf{\Sigma}}_t$):

$$\min \quad J_{SFA}(\mathbf{w}) = \frac{\mathbf{w}^T \langle \dot{\mathbf{x}}\dot{\mathbf{x}}^T \rangle_t \mathbf{w}}{\mathbf{w}^T \langle \mathbf{xx}^T \rangle_t \mathbf{w}} \approx \frac{1}{N} \cdot \frac{\mathbf{w}^T \mathbf{S}_W \mathbf{w}}{\mathbf{w}^T \langle \mathbf{xx}^T \rangle_t \mathbf{w}} - \tilde{p} \cdot \frac{\mathbf{w}^T \tilde{\mathbf{\Sigma}}_t \mathbf{w}}{\mathbf{w}^T \langle \mathbf{xx}^T \rangle_t \mathbf{w}}, \qquad (11)$$

where $N$ is here the total number of points in all trajectories and $\tilde{p}$ is the fraction of transitions between two successive points of the time series that belong to the same trajectory. The weight vector $\mathbf{w}$ which minimizes the first term in (11) is equal to the weight vector found by the application of FLD to the classification problem of the individual trajectory points (note that $\mathbf{S}_B$ enters (11) through $\langle \mathbf{xx}^T \rangle_t$, cf. eq. (9)). The weight vector which maximizes the second term is the one which produces the slowest possible response during individual trajectories. If the separation between the trajectory classes is large compared to the temporal correlations (i.e., the first term in (11) dominates for the resulting $\mathbf{w}$) the slowest feature will be similar to the weight vector found by FLD on the corresponding classification problem. On the other hand, as the temporal correlations of the trajectories increase, i.e., the trajectories themselves become smoother, the slowest feature will tend to favor exploiting this temporal structure of the trajectories over the separation of different classes (in this case, (11) is dominated by the second term for the resulting $\mathbf{w}$).

## 5   Application to linear readouts of a cortical microcircuit model

In the following we discuss several computer simulations of a cortical microcircuit of spiking neurons that demonstrate the theoretical arguments given in the previous section. We trained a number of linear SFA readouts[3] on a sequence of trajectories of network states, each of which is defined by the low-pass filtered spike trains of the neurons in the circuit. Such recurrent circuits typically provide a temporal integration of the input stream and project it nonlinearly into a high-dimensional space [7], thereby boosting the expressive power of the subsequent linear SFA readouts. Note, however, that the optimization (2) implicitly performs an additional whitening of the circuit response. As a model for a cortical microcircuit model we use the laminar circuit from [8] consisting of 560 spiking neurons organized into layers 2/3, 4, and 5, with layer-specific connection probabilities obtained from experimental data [9, 10].

In the first experiment we investigated the ability of SFA to detect a repeating firing pattern within noise input of the same firing statistics. We recorded circuit trajectories in response to 200 repetitions of a fixed spike pattern which are embedded into a continuous Poisson input stream of the same rate. We then trained linear SFA readouts on this sequence of circuit trajectories (we used an exponential

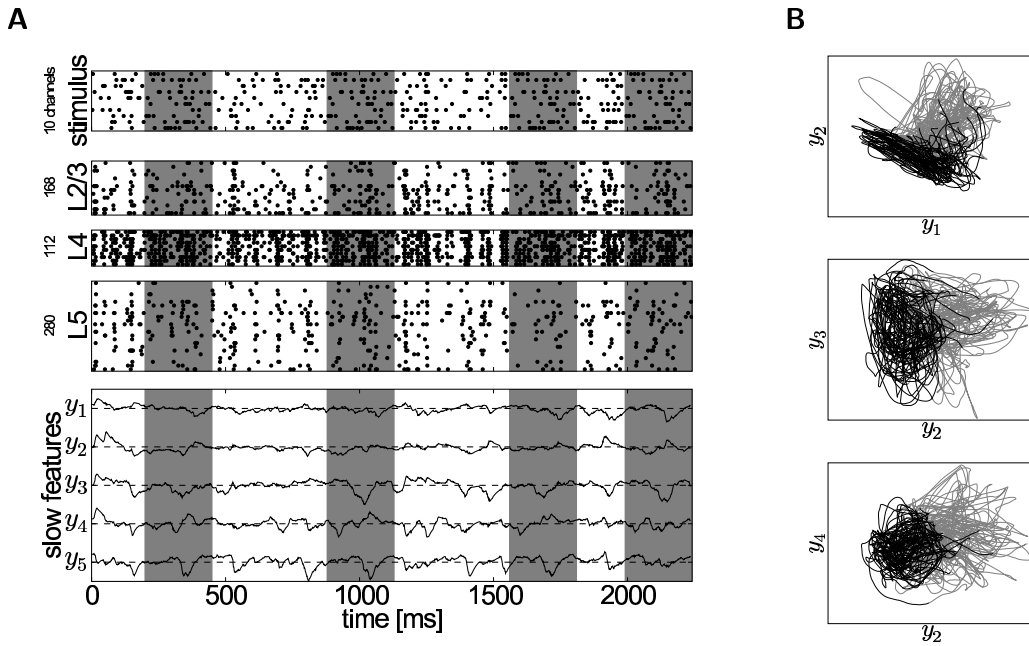

Figure 2: Detecting embedded spike patterns. (**A**) From top to bottom: sample stimulus sequence, response spike trains of the network, and slowest features. The stimulus consists of 10 channels and is defined by repetitions of a fixed spike pattern (dark gray) which are embedded into random Poisson input of the same rate. The pattern has a length of 250ms and is made up by Poisson spike trains of rate 20Hz. The period between two repetitions is drawn uniformly between 100ms and 500ms. The response spike trains of the laminar circuit of [8] are shown separated into layers 2/3, 4, and 5. The numbers of neurons in the layers are indicated on the left, but only the response of every 12th neuron is plotted. Shown are the 5 slowest features, $y_1$ to $y_5$, for the network response shown above. The dashed lines indicate values of 0. (**B**) Phase plots of low-pass filtered versions (leaky integration, $\tau = 100$ms) of individual slow features in response to a test sequence of 50 embedded patterns plotted against each other (black: traces during the pattern, gray: during random Poisson input).

filter with $\tau = 30$ms and a sample time of 1ms). The period of Poisson input in between two such patterns was randomly chosen.

At first glance there is no clear difference in Figure 2A between the raw SFA responses during periods of pattern presentations and during phases of noise input due to the same firing statistics. However, we found that on average the slow feature responses during noise input are zero, whereas a characteristic response remains during pattern presentations. This effect is predicted by the theoretical arguments in section 4. It can be seen in phase plots of traces that are obtained by a leaky integration of the slowest features in response to a test sequence of 50 embedded patterns (see Figure 2B) that the slow features span a subspace where the response during pattern presentations can be nicely separated from the response during noise input. That is, by simple threshold operations on the low-pass filtered versions of the slowest features one can in principle detect the presence of patterns within the continuous input stream. Furthermore, this extracted information is not only available after a pattern has been presented, but already during the presentation of the pattern, which supports the idea of "anytime" computing.

In the second experiment we tested whether SFA is able to discriminate two classes of trajectories as described in section 4. We performed a speech recognition task using the dataset considered originally in [11] and later in the context of biological circuits in [7, 12, 13]. This isolated spoken digits dataset consists of the audio signals recorded from 5 speakers pronouncing the digits "zero", "one", ..., "nine" in ten different utterances (trials) each. We preprocessed the raw audio files with a model of the cochlea [14] and converted the resulting analog cochleagrams into 20 spike trains (using the algorithm in [15]) that serve as input to our microcircuit model (see Figure 3A). We tried to dis-

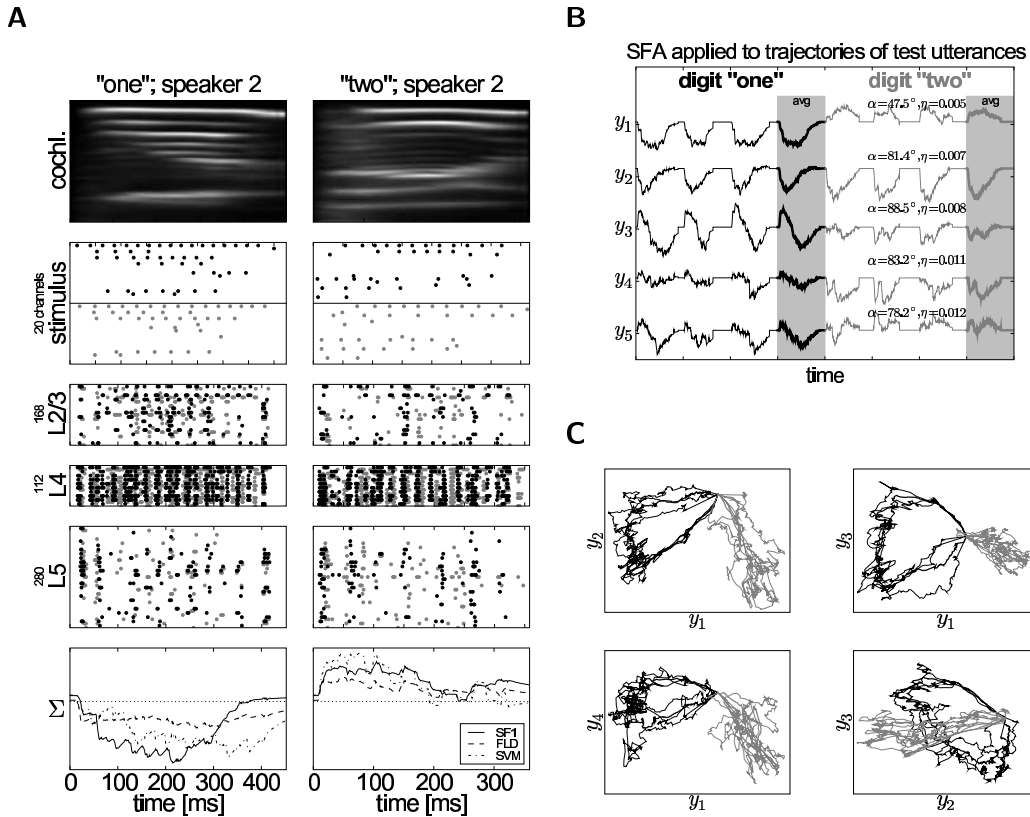

Figure 3: SFA applied to digit recognition of a single speaker. (**A**) From top to bottom: cochlea-grams, input spike trains, response spike trains of the network, and traces of different linear readouts. Each cochleagram has 86 channels with analog values between 0 and 1 (white, near 1; black, near 0). Stimulus spike trains are shown for two different utterances of the given digit (black and gray, the black spike times correspond to the cochleagram shown above). The response spike trains of the laminar circuit from [8] are shown separated into layers 2/3, 4, and 5. The number of neurons in each layer is indicated on the left, but only the response of every 12th neuron is plotted. The responses to the two stimulus spike trains in the panel above are shown superimposed with the corresponding color. Each readout trace corresponds to a weighted sum ($\Sigma$) of network states of the black responses in the panel above. The trace of the slowest feature ("SF1", see **B**) is compared to traces of readouts trained by FLD and SVM with linear kernel to discriminate at any time between the network states of the two classes. All weight vectors are normalized to length 1. The dotted line denotes the threshold of the respective linear classifier. (**B**) Response of the 5 slowest features $y_1$ to $y_5$ of the previously learned SFA in response to trajectories of the three test utterances of each class not used for training (black, digit "one"; gray, digit "two"). The slowness index $\eta = T/2\pi\sqrt{\Delta(y)}$ [2] is calculated from these output signals. The angle $\alpha$ denotes the deviation of the projection direction of the respective feature from the direction found by FLD. The thick curves in the shaded area display the mean SFA responses over all three test trajectories for each class. (**C**) Phase plots of individual slow features plotted against each other (thin lines: individual responses, thick lines: mean response over all test trajectories).

criminate between trajectories in response to inputs corresponding to utterances of digits "one" and "two", of a single speaker. We kept three utterances of each digit for testing and generated from the remaining training samples a sequence of 100 input samples, recorded for each sample the response of the circuit, and concatenated the resulting trajectories in time. Note that here we did not switch the classes of two successive trajectories with a certain probability because, as explained in the previous section, for long trajectories the SFA response is independent of this switching probability. Rather, we trained linear SFA readouts on a completely random trajectory sequence.

Figure 3B shows the 5 slowest features, $y_1$ to $y_5$, ordered by decreasing slowness in response to the trajectories corresponding to the three remaining test utterances for each class, digit "one" and digit "two". In this example, already the slowest feature $y_1$ extracts the class of the input patterns almost perfectly: it responds with positive values for trajectories in response to utterances of digit "two" and with negative values for utterances of digit "one". This property of the extracted features, to respond differently for different stimulus classes, is called the *What*-information [2]. The second slowest feature $y_2$, on the other hand, responds with shapes whose sign is independent of the pattern identity. One can say that, in principle, $y_2$ encodes simply the presence of and the location within a response. This is a typical example of a representation of *Where*-information [2], i.e., the "pattern location" regardless of the identity of the pattern. The other slow features $y_3$ to $y_5$ do not extract either *What*- or *Where*-information explicitly, but rather a mixed version of both. As a measure for the discriminative capability of a specific SFA response, i.e., its quality as a possible classifier, we measured the angle between the projection direction corresponding to this slow feature and the direction of the FLD. It can be seen in Figure 3B that the slowest feature $y_1$ is closest to the FLD. Hence, according to (11), this constitutes an example where the separation between classes dominates, but is already significantly influenced by the temporal correlations of the circuit trajectories.

Figure 3C shows phase plots of these slow features shown in Figure 3B plotted against each other. In the three plots involving feature $y_1$ it can be seen that the directions of the response vector (i.e., the vector composed of the slow feature values at a particular point in time) cluster at class-specific angles, which is characteristic for *What*-information. On the other hand, these phase plots tend to form loops in phase space (instead of just straight lines from the origin), where each point on this loop corresponds to a position within the trajectory. This is a typical property of *Where*-information. Similar responses have been theoretically predicted in [4] and found in simulations of a hierarchical (nonlinear) SFA network trained with a sequence of one-dimensional trajectories [2].

This experiment demonstrates that SFA extracts information about the spoken digit in an unsupervised manner by projecting the circuit trajectories onto a subspace where they are nicely separable so that they can easily be classified by later processing stages. Moreover, this information is provided not only at the end of a specific trajectory, but is made available already much earlier. After sufficient training, the slowest feature $y_1$ in Figure 3B responds with positive or negative values indicating the stimulus class almost during the whole duration of of the network trajectory. This again supports the idea of "anytime" computing. It can be seen in the bottom panel of Figure 3A that the slowest feature, which is obtained in an unsupervised manner, achieves a good separation between the two test trajectories, comparable to the supervised methods of FLD and Support Vector Machine (SVM) [16] with linear kernel.

## 6    Discussion

The results of our paper show that Slow Feature Analysis is in fact a very powerful tool, which is able to approximate the classification capability that results from supervised classification learning. Its elegant formulation as a generalized eigenvalue problem has allowed us to establish a relationship to the supervised method of Fisher's Linear Discriminant (FLD). A more detailed discussion of this relationship, including complete derivations, can be found in [6]. If temporal contiguous points in the time series are likely to belong to the same class, SFA is able to extract the class as a slowly varying feature in an unsupervised manner. This ability is of particular interest in the context of biologically realistic neural circuits because it could enable readout neurons to extract from the trajectories of network states information about the stimulus – without any "teacher", whose existence is highly dubious in the brain. We have shown in computer simulations of a cortical microcircuit model that linear readouts trained with SFA are able to detect specific spike patterns within a stream of spike trains with the same firing statistics and to discriminate between different spoken digits. Moreover, SFA provides in these tasks an "anytime" classification capability.

**Acknowledgments**

We would like to thank Henning Sprekeler and Laurenz Wiskott for stimulating discussions. This paper was written under partial support by the Austrian Science Fund FWF project # S9102-N13 and project # FP6-015879 (FACETS), project # FP7-216593 (SECO) and project # FP7-231267 (ORGANIC) of the European Union.

## Footnotes

[1] A first link between SFA and pattern recognition has been established in [5]. There the optimization is performed over all possible pattern pairs of the same class. However, it might often be implausible to have access to such an artificial time series, e.g., from the perspective of a readout neuron that receives input on-the-fly. We take a different approach and apply the standard SFA algorithm to a time series consisting of randomly selected patterns of the classification problem, where the class at each time step is switched with a certain probability.

[2]In fact, for sufficiently long trajectories the SFA objective becomes effectively independent of the switching probability.

[3]We interpret the linear combination defined by each slow feature as the weight vector of a hypothetical linear readout.

# References

[1] N. Li and J. J. DiCarlo. Unsupervised natural experience rapidly alters invariant object representation in visual cortex. *Science*, 321:1502–1507, 2008.

[2] L. Wiskott and T. J. Sejnowski. Slow feature analysis: unsupervised learning of invariances. *Neural Computation*, 14(4):715–770, 2002.

[3] R. A. Fisher. The use of multiple measurements in taxonomic problems. *Annuals of Eugenics*, 7:179–188, 1936.

[4] L. Wiskott. Slow feature analysis: A theoretical analysis of optimal free responses. *Neural Computation*, 15(9):2147–2177, 2003.

[5] P. Berkes. Pattern recognition with slow feature analysis. Cognitive Sciences EPrint Archive (CogPrint) 4104, February 2005. http://cogprints.org/4104/.

[6] S. Klampfl and W. Maass. A theoretical basis for emergent pattern discrimination in neural systems through slow feature extraction. Submitted for publication, 2009.

[7] W. Maass, T. Natschläger, and H. Markram. Real-time computing without stable states: A new framework for neural computation based on perturbations. *Neural Computation*, 14(11):2531–2560, 2002.

[8] S. Häusler and W. Maass. A statistical analysis of information processing properties of lamina-specific cortical microcircuit models. *Cerebral Cortex*, 17(1):149–162, 2007.

[9] A. Gupta, Y. Wang, and H. Markram. Organizing principles for a diversity of GABAergic interneurons and synapses in the neocortex. *Science*, 287:273–278, 2000.

[10] A. M. Thomson, D. C. West, Y. Wang, and A. P. Bannister. Synaptic connections and small circuits involving excitatory and inhibitory neurons in layers 2–5 of adult rat and cat neocortex: triple intracellular recordings and biocytin labelling in vitro. *Cerebral Cortex*, 12(9):936–953, 2002.

[11] J. J. Hopfield and C. D. Brody. What is a moment? Transient synchrony as a collective mechanism for spatio-temporal integration. *Proc. Nat. Acad. Sci. USA*, 98(3):1282–1287, 2001.

[12] D. Verstraeten, B. Schrauwen, D. Stroobandt, and J. Van Campenhout. Isolated word recognition with the liquid state machine: a case study. *Inf. Process. Lett.*, 95(6):521–528, 2005.

[13] R. Legenstein, D. Pecevski, and W. Maass. A learning theory for reward-modulated spike-timing-dependent plasticity with application to biofeedback. *PLoS Computational Biology*, 4(10):1–27, 2008.

[14] R. F. Lyon. A computational model of filtering, detection, and compression in the cochlea. In *Proc. IEEE Int. Conf. Acoustics Speech and Signal Processing*, pages 1282–1285, May 1982.

[15] B. Schrauwen and J. V. Campenhout. BSA, a fast and accurate spike train encoding scheme. In *Proceedings of the International Joint Conference on Neural Networks*, 2003.

[16] B. Schölkopf and A. J. Smola. *Learning with Kernels*. MIT Press, Cambridge, MA, 2002.

